# An Analog Neural Network Inspired by Fractal Block Coding

**Fernando J. Pineda**
The Applied Physics Laboratory
The Johns Hopkins University
Johns Hokins Road
Laurel, MD 20723-6099

**Andreas G. Andreou**
Dept. of Electrical & Computer
Engineering
The Johns Hopkins University
34th & Charles St.
Baltimore, MD 21218

## Abstract

We consider the problem of decoding block coded data, using a physical dynamical system. We sketch out a decompression algorithm for fractal block codes and then show how to implement a recurrent neural network using physically simple but highly-nonlinear, analog circuit models of neurons and synapses. The nonlinear system has many fixed points, but we have at our disposal a procedure to choose the parameters in such a way that only one solution, the desired solution, is stable. As a partial proof of the concept, we present experimental data from a small system a 16-neuron analog CMOS chip fabricated in a 2m analog p-well process. This chip operates in the subthreshold regime and, for each choice of parameters, converges to a unique stable state. Each state exhibits a qualitatively fractal shape.

## 1. INTRODUCTION

Sometimes, a nonlinear approach is the simplest way to solve a linear problem. This is true when computing with physical dynamical systems whose natural operations are nonlinear. In such cases it may be expensive, in terms of physical complexity, to linearize the dynamics. For example in neural computation active ion channels have highly non linear input-output behaviour (see Hille 1984). Another example is

subthreshold CMOS VLSI technology[1]. In both examples the physics that governs the operation of the active devices, gives rise to gain elements that have exponential transfer characteristics. These exponentials result in computing structures with non-linear dynamics. It is therefore worthwhile, from both scientific and engineering perspectives, to investigate the idea of analog computation by highly non-linear components.

This paper, explores an approach for solving a specific linear problem with analog circuits that have nonlinear transfer functions. The computational task considered here is that of fractal block code decompression (see e.g. Jacquin, 1989).

The conventional approach to decompressing fractal codes is essentially an excercise in solving a high-dimenional sparse linear system of equations by using a relaxation algorithm. The relaxation algorithm is performed by iteratively applying an affine transformation to a state vector. The iteration yields a sequence of state vectors that converges to a vector of decoded data. The approach taken in this paper is based on the observation that one can construct a physically-simple nonlinear dyanmical system whose unique stable fixed point coincides with the solution of the sparse linear system of equations.

In the next section we briefly summarize the basic ideas behind fractal block coding. This is followed by a description of an analog circuit with physically-simple nonlinear neurons. We show how to set the input voltages for the network so that we can program the position of the stable fixed point. Finally , we present experimental results obtained from a test chip fabricated in a 2mm CMOS process.

## 2. FRACTAL BLOCK CODING IN A NUTSHELL

Let the N-dimensional state vector $I$ represent a one dimensional curve sampled on N points. An *affine* transformation of this vector is simply a transformation of the form $I'$ $= WI+B$ , where $W$ is an $NxN$ -element matrix and $B$ is an $N$-component vector. This transformation can be iterated to produce a sequence of vectors $I^{(0)},...,I^{(n)}$. The sequence converges to a unique final state $I^*$ that is independent of the initial state $I^{(0)}$ if the maximum eigenvalue $\lambda_{max}$ of the matrix $W$ satisfies $\lambda_{max} <1$. The uniqueness of the final state implies that to transmit the state $I^*$ to a receiver, we can either transmit $I^*$ directly, or we can transmit $W$ and $B$ and let the receiver perform the iteration to generate $I^*$. In the latter case we say that $W$ and $B$ constitute an *encoding* of the state $I^*$. For this encoding to be useful, the amount of data needed to transmit $W$ and $B$ must be less than the amount of data needed to transmit $I^*$ This is the case when $W$ and $B$ are sparse and parameterized and when the total number of bits needed to transmit these parameters is less than the total number of bits needed to transmit the uncompressed state $I^*$

Fractal block coding is a special case of the above approach. It amounts to choosing a

blocked structure for the matrix $W$. This structure forces large-scale features to be mapped into small-scale features. The result is a steady state $I^*$ that represents a curve with self similar (actually self affine) features. As a concrete example of such a structure, consider the following transformation of the state $I$.

$$I'_i = w_L I_{2i+1} + b_L \quad for \quad 0 \le i \le \frac{N}{2} - 1$$

$$I'_i = w_R I_{2i-N} + b_R \quad for \quad \frac{N}{2} \le i \le N-1 \tag{1}$$

This transformation has two blocks. The transformation of the first N/2 components of $I$ depend on the parameters $w_L$ and $b_L$ while the transformation of the second N/2 components depend on the parameters $w_R$, and $b_R$. Consequently just four parameters completely specify this transformation. This transformation can be expressed as a single affine transformation as follows:

$$
\begin{pmatrix} I'_0 \\ \dots \\ I'_{N/2-1} \\ I'_{N/2} \\ \dots \\ I'_{N-1} \end{pmatrix}
=
\begin{pmatrix} & & w_L & & \\ w_R & & & & \\ & & & & w_L \\ & & & w_R & \end{pmatrix}
\begin{pmatrix} I_0 \\ \dots \\ I_{N/2-1} \\ I_{N/2} \\ \dots \\ I_{N-1} \end{pmatrix}
+
\begin{pmatrix} b_L \\ \dots \\ b_L \\ b_R \\ \dots \\ b_R \end{pmatrix}
\tag{2}
$$

The top and bottom halves of $I'$ depend on the odd and even components of $I$ respectively. This subsampling causes features of size $l$ to be mapped into features of size $l/2$. A subsampled copy of the state $I$ with transformed intensities is copied into the top half of $I'$. Similarly, a subsampled copy of the state $I$ with transformed intensities is copied into the bottom half of $I'$. If this transformation is iterated, the sequence of transformed vectors will converge provided the eigenvalues determined by $w_L$ and $w_R$ are all less than one (i.e. $w_L$ and $w_R < 1$).

Although this toy example has just four free parameters and is thus too trivial to be useful for actual compression applications, it does suffice to generate state vectors with fractal properties since at steady state, the top and bottom halves of $I'$ differ from the entire curve by an affine transformation.

In this paper we will not describe how to solve the inverse problem which consists of finding a parameterized affine transformation that produces a given final state $T$. We note, however, that it is a special (and simpler) case of the recurrent network training problem, since the problem is linear, has no hidden units and has only one fixed point. The reader is refered to (Pineda, 1988) or. for a least squares algorithm in the context of neural nets or to (Monroe and Dudbridge, 1992) for a least squares algorithm in the context of coding.

## 3. A CMOS NEURAL NETWORK MODEL

Now that we have described the salient aspects of the fractal decompression problem, we turn to the problem of implementing an analog neural network whose nonlinear dynamics converges to the same fixed point as the linear system. Nonlinearity arises because we

make no special effort to linearize the gain elements (controlled conductances and transconductances) of the implementation medium. In this section we first describe a simple neuron. Then we analyze the dynamics of a network composed of such neurons. Finally we describe how to program the fixed point in the actual physical network.

## 3.1 The analog Neuron

We would like to create a neuron model that calculates the transformation $I^{(out)} = aI^{(in)} + b$. Consider the circuit shown in figure 1. This has three functional sections which compute by adding and subtracting currents and where voltages are ``log'' coded; this is the essence of the ``current-mode'' aproach in circuit design (Andreou et.al. 1994). The first section, receives an input voltage from a presynaptic neuron, converts it into a current I(in), and multiplies it by a weight a. The second section adds and subtracts the bias current b. The last section converts the output current into an output voltage and transmits it to the next neuron in the network. Since the transistors have exponential transfer characteristics, this voltage is logarithmically coded.

The parameters $a$ and $b$ are set by external voltages. The parameter $a$, is set by a single external voltage $v_a$ while the bias parameter $b = b^{(-)} - b^{(+)}$ is set by two external voltages $v_{b^{(+)}}$ and $v_{b^{(-)}}$. Two voltages are used for $b$ to account for both positive and negative bias values since $b^{(-)}>0$ and $b^{(+)}>0$.

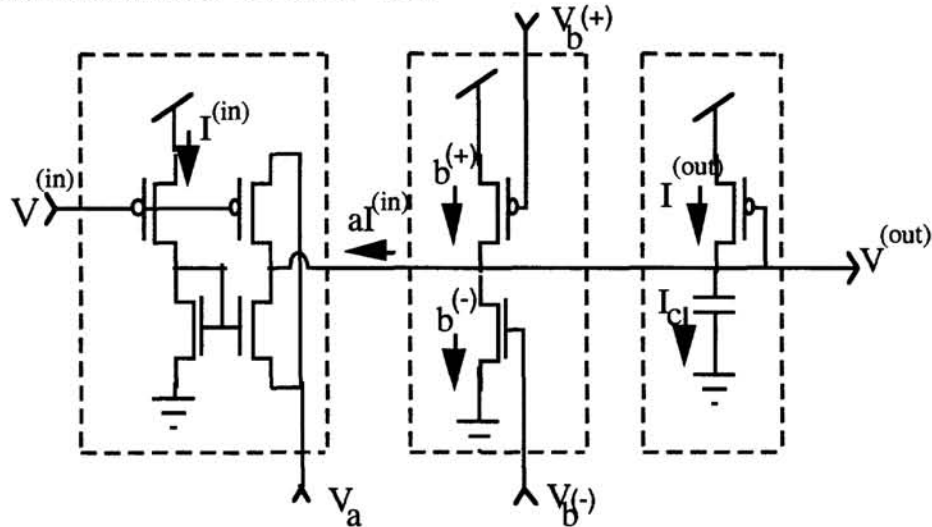

Figure 1. The analog neuron has three sections.

To derive the dynamical equations of the neuron, it is neccesary to add up all the currents and invoke Kirchoff's current law, which requires that

$$I^{(out)} - aI^{(in)} + b^{(+)} - b^{(-)} = I_c. \qquad (3)$$

If we now assume a simple subthreshold model for the behavior of the FET's and PFETs in the neuron, we can obtain the following expression for the current across the capacitor:

$$-\frac{Q}{I^{(out)}} \frac{dI^{(out)}}{dt} = I_c \qquad (4)$$

where $Q = C/\kappa V_{th}$ determines the characteristic time scale of the neuron[2]. It immediately follows from the last two expressions that the dynamics of a single neuron is determined by the equation

$$Q\frac{dI^{(out)}}{dt} = -I^{(out)}(I^{(out)} - aI^{(in)} - b).$$ (5)

Where $b = b^{(-)} - b^{(+)}$. This equation appears to have a quadratic nonlinearity on the r.h.s. In fact, the noninearity is even more complicated since, the cooeficients $a$, $b^{(+)}$ and $b^{(-)}$ are not constants, but depend on $I^{(out)}$ (through $v^{(out)}$). Application of the simple subthreshold model, results in a multiplier gain that is a function of $V^{(out)}$ (and hence $I^{(out)}$) as well as $V_a$. It is given by

$$a\left(v_a, v^{(out)}\right) = 2\exp(-\frac{v_{dd}}{2})\left[\sinh(\frac{v_{dd}}{2} - v_a) - \sinh(\frac{v_{dd}}{2} - v^{(out)})\right].$$ (6)

Similarly, the currents $b^{(+)}$ and $b^{(-)}$ are given by

$$b^{(+)} = I_o^{(fpet)}\exp(\kappa\hat{v}_{b^{(+)}})\left(1 - \exp(-\hat{v}^{(out)})\right)$$ (7.a)

and

$$b^{(-)} = I_o^{(nfet)}\exp(\kappa v_{b^{(-)}})\left(1 - \exp(-v^{(out)})\right)$$ (7.b)

respectively, where $\hat{v}_\alpha \equiv vdd - v_\alpha$.

## 3.2 Network dynamics and Stability considerations

With these results we conclude that, a network of neurons, in which each neuron receives input from only one other neuron, would have a dynamical equation of the form

$$Q\frac{dI_i}{dt} = -I_i(I_i - a_i(I_i)I_{j(i)} - b_i)$$ (8)

where the connectivity of the network is determined by the function $j(i)$. The fixed points of these highly nonlinear equations occur when the r.h.s. of (8) vanishes. This can only happen if either $I_i = 0$ or if $(I_i - a_iI_{j(i)} - b_i) = 0$ for each $i$. The local stability of each of these fixed points follows by examining the eigenvalues ($\lambda$) of the corresponding jacobian. The expression for the jacobian at a general point $I$ is

$$J_{ik} = \frac{\partial F_i}{\partial I_k} = -Q\left[(I_i - a_iI_{j(i)} - b_i)\delta_{ik} + I_i(1 - a_i'I_{j(i)} - b_i')\delta_{ik} - a_iI_i\delta_{j(i)k}\right].$$ (9)

Where the partial derivatives, $a'_i$ and $b'_i$ are with respect to $I_i$. At a fixed point the jacobian takes the form

$$J_{ik} = Q\begin{cases}b_i\delta_{ik} & if \quad I_i = 0 \\ -I_i\left[(1 - a_i'I_{j(i)} - b_i')\delta_{ik} - a_i\delta_{j(i)k}\right] & if \quad (I_i - a_iI_{j(i)} - b_i) = 0\end{cases}.$$ (10)

There are two cases of interest. The first case is when no neurons have zero output. This is the "desired solution." In this case, the jacobian specializes to

$$J_{ik} = -QI_i\left[(1 - a_i'I_{j(i)} - b_i')\delta_{ik} - a_i\delta_{j(i)k}\right].\qquad(11)$$

Where, from (6) and (7), it can be shown that the partial derivatives, $a_i'$ and $b_i'$ are both non-positive. It immediately follows, from Gerschgorin's theorem, that a sufficient condition that the eigenvalues be negative and that the fixed point be stable, is that $|a_i| < 1$. The second case is when at least one of the neurons has zero output. We call these fixed points the "spurious solutions." In this case some of the eigenvalues are very easy to calculate because terms of the form $(b_i - \lambda)$, where $I_i = 0$, can be factored from the expression for $\det(J - \lambda I)$. Thus some eigenvalues can be made positive by making some of the $b_i$ positive. Accordingly, if all the $b_i$ satisfy $b_i > 0$, some of the eigenvalues will necessarily be positive and the spurious solutions will be unstable. To summarize the above discussion, we have shown that by choosing $b_i > 0$ and $|a_i| < 1$ for all $i$, we can make the desired fixed point stable and the spurious fixed points unstable. Note that a sufficient condition for $b_i > 0$ is if $b_i^{(+)} = 0$.

It remains to show that the system must converge to the desired fixed point, i.e. that the system cannot oscillate or wander chaotically. To do this we consider the connectivity of the network we implemented in our test chip. This is shown schematically in figure 2. The first eight neurons receive input from the odd numbered neurons while the second eight neurons receive input from the even numbered neurons. The neurons on the left-hand side all share the weight, $w_L$, while the neurons on the right share the weight $w_R$. By tracing the connections, we find that there are two independent loops of neurons: loop #1 = {0,8,12,14,15,7,3,1} and loop #2 = {2,9,4,10,13,6,11,5}.

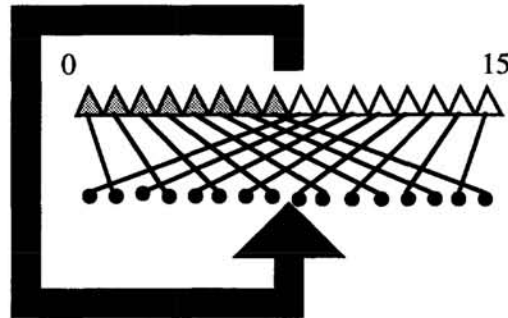

Figure 2. The connection topology for the test chip is determined by the matrix of equation (1). The neurons are labeled 0-15.

By inspecting each loop, we see that it passes through either the left or right hand range an even number of times. Hence, if there are any inhibitory weights in a loop, there must be an even number of them. This is the "even loop criterion", and it suffices to prove that the network is globally asymptotically stable, (Hirsch, 1987).

## 3.3. Programming the fixed point

The nonlinear circuit of the previous section converges to a fixed point which is the solution of the following system of transcendental equations

$$I_i^* - a_i(I_i^*, v_a)I_{j(i)}^* - b_i^{(-)}(I_i^*, v_{b^{(-)}}) = 0\qquad(12)$$

where the coefficients $a_i$ and $b_i$ are given by equations (6) and (7b) respectively. Similarly, the iterated affine transformations converge to the solution of the following linear equations

$$I_i^* - A_i I_{j(i)}^* - B_i = 0 \tag{13}$$

where the coefficients $\{A_i, B_i\}$ and the connections $j(i)$ are obtained by solving the approximate inverse problem with the additional constraints that $b_i > 0$ and $|a_i| < 1$ for all $i$,. The requirement that the fixed points of the two systems be identical results in the conditions

$$
\begin{aligned}
A_i &= a_i(I_i^*, v_a) \\
B_i &= b_i^{(-)}(I_i^*, v_{b^{(-)}})
\end{aligned}
\tag{14}
$$

These equations can be solved for the required input voltages $v_a$, and $v_{b^{(-)}}$. Thus we are able to construct a nonlinear dynamical system that converges to the same fixed point as a linear system. For this programming method to work, of course, the subthreshold model we have used to characterize the network must accurately model the physical properties of the neural network.

## 4. PRELIMINARY RESULTS

As a first step towards realizing a working system, we fabricated a Tiny chip containing 16 neurons arranged in two groups of eight. The topology is the same as shown in figure 2. The neurons are similar to those in figure 1 except that the bias term in each block of 8 neurons has the form $b = kb^{(-)} + (7-k)\bar{b}^{(-)}$, where $0 \leq k \leq 7$ is the label of a particular neuron within a block. This form increases the complexity of the neurons, but also allows us to represent ramps more easily (see figure 3).

We fabricated the chip through MOSIS in a 2μm p-well CMOS process. A switching layer allows us to change the connection topology at run-time. One of the four possible configurations corresponds to the toplogy of figure 2. Six external voltages $\{v_{a_L}, v_{b^{(-)}}, v_{\bar{b}^{(-)}}, v_{a_R}, v_{b^{(-)}}, v_{\bar{b}^{(-)}}\}$ parameterize the fixed points of the network. These are controlled by potentiometers. There is multiplexing circuitry included on the chip that selects which neuron output is to be amplified by a sense-amp and routed off-chip. The neurons can be addressed individually by a 4-bit neuron address. The addressing and analog-to-digital conversion is performed by a Motorolla 68HC11A1 microprocessor.

We have operated the chip at 5volts and at 2.6 volts. Figure 3. shows the scanned steady state output of one of the test chips for a particular choice of input parameters with $v_{dd} = 5$ volts. The curve in figure 3. exhibits the qualitatively self-similar features of a recursively generated object. We are able to see three generations of a ramp. At 2.5 volts we see a very similar curve. We find that the chip draws 16.3 μA at 2.5 volts. This corresponds to a steady state power dissipation of 41μW. Simulations indicate that the chip is operating in the subthreshold regime when $v_{dd} = 2.5$ volts. Simulations also indicate that the chip settles in less than one millisecond. We are unable to perform quantitiative measurements with the first chip because of several layout errors. On the other hand, we have experimentally verified that the network is indeed stable and that network produces qualitative fractals. We explored the parameter space informatlly. At no time did we encounter anything but the desired solutions.

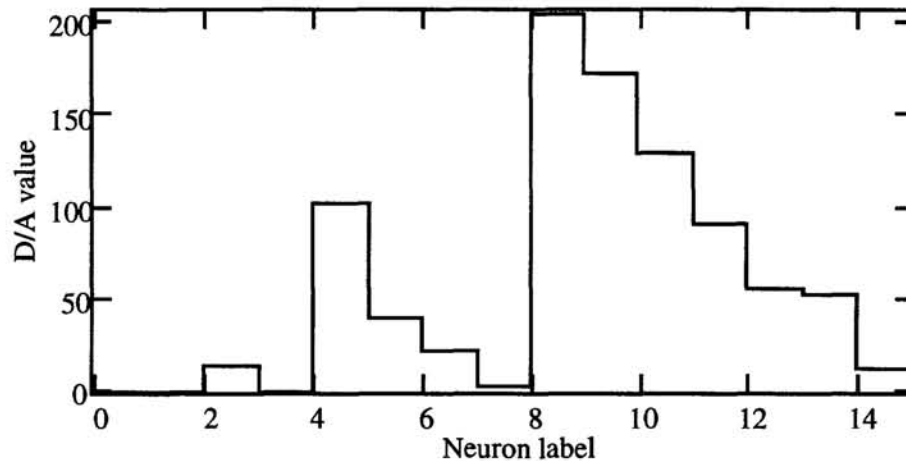

Figure 3 D/A output for chip #3 for a particular set of input voltages.

We have already fabricated a larger design without the layout problems of the prototype. This second design has 32 pixels and a richer set of permitted topologies. We expect to make quantitative measurements with this second design. In particular we hope to use it to decompress an actual block code.

**Acknowledgements**

The work described here is funded by APL IR&D as well as a grant from the National Science Foundation ECS9313934, Paul Werbos is the monitor. The authors would like to thank Robert Jenkins , Kim Strohbehn and Paul Furth for many useful conversations and suggestions.

**References**

Andreou, A.G. and Boahen, K.A. Neural Information Processing I: The Current-Mode approach, *Analog VLSI: Signal and Information Processing*, (eds: M Ismail and T. Fiez) MacGraw-Hill Inc., New York. Chapter 6 (1994).

Hille, B., *Ionic Channels of Excitable Membranes*, Sunderland, MA, Sinauer Associates Inc. (1984).

Hirsch, M. ,Convergence in Neural Nets, *Proceedings of the IEEE ICNN*, San Diego, CA, (1987).

Jacquin , A. E., *A Fractal Theory of iterated Markov operators with applications to digital image coding*, Ph.D. Dissertation, Georgia Institute of Technology (1989).

Mead, C., *Analog VLSI and Neural Systems*, Addison Wesley, (1989)

Monroe, D.M. and Dudbridge, F. Fractal block coding of images, *Electronics Letters*, **28**, pp. 1053-1055, (1992).

Pineda, F.J., Dynamics and Architecuture for Neural Computation, *Journal of Complexity*, **4**, 216-245 (1988).

## Footnotes

[1]We consider subthreshold analog VLSI., (Mead 1989; Andreou and Boahen, 1994). A simple subthreshold model is $I_{ds} = I_o^{(nfet)} \exp(\kappa v_{gb})(\exp(-v_{sb}) - \exp(-v_{db}))$ for NFETS, where $\kappa \sim 0.67$ and $I_o^{(nfet)} = 9.7 \times 10^{-18}$ A. The voltage differences $v_{gb}$, $v_{sb}$, and $v_{db}$ are in units of the thermal voltage, $V_{th}= 0.025$V. We use a corresponding expression for PFETs of the from $I_{ds} = I_o^{(pfet)} \exp(-\kappa v_{gb})(\exp(v_{sb}) - \exp(v_{db}))$ where $I_o^{(pfet)} = 3.8 \times 10^{-18}$ A.

[2]C represents the total gate capacitance from all the transistors connected to the horizontal line of the neuron. For the $2\mu$ analog process, the gate capacitance is approximately 0.5 fF/$\mu^2$ so a $10\mu$ x $10\mu$ FET has a characteristic charge of Q $=2.959$ x $10^{-14}$ Coulombs at room temperature.
